# A BIFURCATION THEORY APPROACH TO THE PROGRAMMING OF PERIODIC ATTRACTORS IN NETWORK MODELS OF OLFACTORY CORTEX

Bill Baird
Department of Biophysics
U.C. Berkeley

## ABSTRACT

A new learning algorithm for the storage of static and periodic attractors in biologically inspired recurrent analog neural networks is introduced. For a network of n nodes, n static or n/2 periodic attractors may be stored. The algorithm allows programming of the network vector field independent of the patterns to be stored. Stability of patterns, basin geometry, and rates of convergence may be controlled. For orthonormal patterns, the learning operation reduces to a kind of periodic outer product rule that allows local, additive, commutative, incremental learning. Standing or traveling wave cycles may be stored to mimic the kind of oscillating spatial patterns that appear in the neural activity of the olfactory bulb and prepyriform cortex during inspiration and suffice, in the bulb, to predict the pattern recognition behavior of rabbits in classical conditioning experiments. These attractors arise, during simulated inspiration, through a multiple Hopf bifurcation, which can act as a critical "decision point" for their selection by a very small input pattern.

## INTRODUCTION

This approach allows the construction of biological models and the exploration of engineering or cognitive networks that employ the type of dynamics found in the brain. Patterns of 40 to 80 hz oscillation have been observed in the large scale activity of the olfactory bulb and cortex(Freeman and Baird 86) and even visual neocortex(Freeman 87,Grey and Singer 88), and found to predict the olfactory and visual pattern recognition responses of a trained animal. Here we use analytic methods of bifurcation theory to design algorithms for determining synaptic weights in recurrent network architectures, like those

found in olfactory cortex, for associative memory storage of
these kinds of dynamic patterns.

The "projection algorithm" introduced here employs higher
order correlations, and is the most analytically transparent
of the algorithms to come from the bifurcation theory ap-
proach(Baird 88). Alternative numerical algorithms employing
unused capacity or hidden units instead of higher order corr-
elations are discussed in (Baird 89). All of these methods
provide solutions to the problem of storing exact analog at-
tractors, static or dynamic, in recurrent neural networks, and
allow programming of the ambient vector field independent of
the patterns to be stored. The stability of cycles or equi-
libria, geometry of basins of attraction, rates of convergence
to attractors, and the location in parameter space of primary
and secondary bifurcations can be programmed in a prototype
vector field - the normal form.

To store cycles by the projection algorithm, we start with the
amplitude equations of a polar coordinate normal form, with
coupling coefficients chosen to give stable fixed points on
the axes, and transform to Cartesian coordinates. The axes of
this system of nonlinear ordinary differential equations are
then linearly transformed into desired spatial or spatio-tem-
poral patterns by projecting the system into network coordina-
tes - the standard basis - using the desired vectors as colum-
ns of the transformation matrix. This method of network syn-
thesis is roughly the inverse of the usual procedure in bifur-
cation theory for analysis of a given physical system.

Proper choice of normal form couplings will ensure that the
axis attractors are the only attractors in the system - there
are no "spurious attractors". If symmetric normal form coef-
ficients are chosen, then the normal form becomes a gradient
vector field. It is exactly the gradient of an explicit poten-
tial function which is therefore a strict Liapunov function
for the system. Identical normal form coefficients make the
normal form vector field equivariant under permutation of the
axes, which forces identical scale and rotation invariant
basins of attraction bounded by hyperplanes. Very complex
periodic attractors may be established by a kind of Fourier
synthesis as linear combinations of the simple cycles chosen
for a subset of the axes, when those are programmed to be
unstable, and a single "mixed mode" in the interior of that
subspace is made stable. Proofs and details on vectorfield
programming appear in (Baird 89).

In the general case, the network resulting from the projection

algorithm has fourth order correlations, but the use of restrictions on the detail of vector field programming and the types of patterns to be stored result in network architectures requiring only second order correlations. For biological modeling, where possibly the patterns to be stored are sparse and nearly orthogonal, the learning rule for periodic patterns becomes a "periodic" outer product rule which is local, additive, commutative, and incremental. It reduces to the usual Hebb-like rule for static attractors.

## CYCLES

The observed physiological activity may be idealized mathematically as a "cycle", $r \, x_j \, e^{i(\theta j + wt)}$ , $j=1,2,...,n$. Such a cycle is a "periodic attractor" if it is stable. The global amplitude r is just a scaling factor for the pattern $\underline{x}$ , and the global phase w in $e^{iwt}$ is a periodic scaling that scales $\underline{x}$ by a factor between ± 1 at frequency w as t varies.

The same vector $\underline{x}^s$ or "pattern" of relative amplitudes can appear in space as a standing wave, like that seen in the bulb, if the relative phase $\theta^s_i$ of each compartment (component) is the same, $\theta^s_{i+1} = \theta^s_i$, or as a traveling wave, like that seen in the prepyriform cortex, if the relative phase components of $\underline{\theta}^s$ form a gradient in space, $\theta^s_{i+1} = 1/\alpha \, \theta^s_i$. The traveling wave will "sweep out" the amplitude pattern $\underline{x}^s$ in time, but the root-mean-square amplitude measured in an experiment will be the same $\underline{x}^s$, regardless of the phase pattern. For an arbitrary phase vector, these "simple" single frequency cycles can make very complicated looking spatio-temporal patterns. From the mathematical point of view, the relative phase pattern $\underline{\theta}$ is a degree of freedom in the kind patterns that can be stored. Patterns of uniform amplitude $\underline{x}$ which differed only in the phase locking pattern $\underline{\theta}$ could be stored as well.

To store the kind of patterns seen in bulb, the amplitude vector $\underline{x}$ is assumed to be parsed into equal numbers of excitatory and inhibitory components, where each class of component has identical phase, but there is a phase difference of 60 – 90 degrees between the classes. The traveling wave in the prepyriform cortex is modeled by introducing an additional phase gradient into both excitatory and inhibitory classes.

## PROJECTION ALGORITHM

The central result of this paper is most compactly stated as the following:

## THEOREM

Any set S, s = 1,2, ...,n/2 , of cycles  $r^s x_j^s e^{i(\theta js + wst)}$  of linearly independent vectors of relative component amplitudes $\underline{x}^s \in R^n$ and phases $\underline{\theta}^s \in S^n$, with frequencies $w^s \in R$ and global amplitudes $r^s \in R$, may be established in the vector field of the analog fourth order network:

$$\dot{x}_i = -\tau x_i + E_j T_{ij} x_j + E_{jkl} T_{ijkl} x_j x_k x_l + b_i \delta(t)$$

by some variant of the projection operation :

$$T_{ij} = E_{mn} P_{im} J_{mn} P^{-1}_{nj} , \quad T_{ijkl} = E_{mn} P_{im} A_{mn} P^{-1}_{mj} P^{-1}_{nk} P^{-1}_{nl} ,$$

where the n x n matrix P contains the real and imaginary components $[\underline{x}^s \cos \underline{\theta}^s , \underline{x}^s \sin \underline{\theta}^s]$ of the complex eigenvectors $\underline{x}^s e^{i\theta s}$ as columns, J is an n x n matrix of complex conjugate eigenvalues in diagonal blocks, $A_{mn}$ is an n x n matrix of 2x2 blocks of repeated coefficients of the normal form equations, and the input $b_i \delta(t)$ is a delta function in time that establishes an initial condition. The vector field of the dynamics of the global amplitudes $r_s$ and phases $\phi_s$ is then given exactly by the normal form equations :

$$\dot{r}_s = u_s r_s - r_s E_j a_{sj} r_j^2$$

$$\dot{\phi}_s = w_s + E_j b_{sj} r_j^2$$

In particular, for $a_{sk} > 0$ , and $a_{ss}/a_{ks} < 1$ , for all s and k, the cycles  s = 1,2,...,n/2  are stable, and have amplitudes $r_s = (u_s/a_{ss})^{1/2}$, where  $u_s = 1 - \tau$ .

Note that there is a multiple Hopf bifurcation of codimension n/2 at $\tau = 1$. Since there are no approximations here, however, the theorem is not restricted to the neighborhood of this bifurcation, and can be discussed without further reference to bifurcation theory. The normal form equations for $dr^s/dt$ and $d\phi^s/dt$ determine how $r^s$ and $\phi^s$ for pattern s evolve in time in interaction with all the other patterns of the set S. This could be thought of as the process of phase locking of the pattern that finally emerges. The unusual power of this algorithm lies in the ability to precisely specify these non-linear interactions. In general, determination of the modes of the linearized system alone (Li and Hopfield 89) is insufficient to say what the attractors of the nonlinear system will be.

## PROOF

The proof of the theorem is instructive since it is a constructive proof, and we can use it to explain the learning algorithm. We proceed by showing first that there are always fixed points on the axes of these amplitude equations, whose stability is given by the coefficients of the nonlinear terms. Then the network above is constructed from these equations by two coordinate transformations. The first is from polar to Cartesian coordinates, and the second is a linear transformation from these canonical "mode" coordinates into the standard basis $e_1$, $e_2$, ..., $e_N$, or "network coordinates". This second transformation constitutes the "learning algorithm", because it transforms the simple fixed points of the amplitude equations into the specific spatio-temporal memory patterns desired for the network.

### Amplitude Fixed Points

Because the amplitude equations are independent of the rotation $\phi$, the fixed points of the amplitude equations characterize the asymptotic states of the underlying oscillatory modes. The stability of these cycles is therefore given by the stability of the fixed points of the amplitude equations. On each axis $r_s$, the other components $r_j$ are zero, by definition,

$$\dot{r}_j = r_j ( u_j - E_k a_{jk} r_k^2 ) = 0 , \quad \text{for } r_j = 0 , \quad \text{which leaves}$$

$$\dot{r}_s = r_s ( u_s - a_{ss} r_s^2 ) , \quad \text{and} \quad \dot{r}_s = 0 , \quad \text{when} \quad r_s^2 = u_s/a_{ss} .$$

There is an equilibrium on each axis s, at $r_s = (u_s/a_{ss})^{1/2}$, as claimed. Now the Jacobian of the amplitude equations at some fixed point $r\hat{}$ has elements

$$J_{ij} = - 2 a_{ij} r\hat{}_i r\hat{}_j , \quad J_{ii} = u_i - 3 a_{ii} r\hat{}_i^2 - \overset{n}{\underset{j \neq i}{E}} a_{ij} r\hat{}_j^2 .$$

For a fixed point $r\hat{}_s$ on axis s, $J_{ij} = 0$ , since $r\hat{}_i$ or $r\hat{}_j = 0$, making J a diagonal matrix whose entries are therefore its eigenvalues. Now $J_{ii} = u_i - a_{is} r\hat{}_s^2$, for i $\neq$ s, and $J_{ss} = u_s - 3 a_{ss} r\hat{}_s^2$. Since $r\hat{}_s^2 = u_s/a_{ss}$, $J_{ss} = - 2 u_s$, and $J_{ii} = u_i - a_{is} (u_s/a_{ss})$. This gives $a_{is}/a_{ss} > u_i/u_s$ as the condition for negative eigenvalues that assures the stability of $r\hat{}_s$. Choice of $a_{ji}/a_{ii} > u_j/u_i$, for all i,j , therefore guarantees stability of all axis fixed points.

### Coordinate Transformations

We now construct the neural network from these well behaved equations by the following transformations,

First; polar to Cartesian , $(r_s, \phi_s)$ to $(v_{2s-1}, v_{2s})$ : Using

$$v_{2s-1} = r_s \cos \phi_s , \quad v_{2s} = r_s \sin \phi_s , \quad \text{and differentiating these}$$

gives:

$$\dot{v}_{2s-1} = \dot{r}_s \cos \phi_s - r_s \sin \phi_s \, \dot{\phi}_s \ ,$$

$$\dot{v}_{2s} = \dot{r}_s \sin \phi_s + r_s \cos \phi_s \, \dot{\phi}_s \ ,$$

by the chain rule. Now substituting $\cos \phi_s = v_{2s-1}/r_s$ , and $r_s \sin \phi_s = v_{2s}$,

gives:

$$\dot{v}_{2s-1} = (v_{2s-1}/r_s) \, \dot{r}_s - v_{2s} \, \dot{\phi}_s$$

$$\dot{v}_{2s} = v_{2s} \, \dot{r}_s + (v_{2s-1}/r_s) \, \dot{\phi}_s$$

Entering the expressions of the normal form for $\dot{r}_s$ and $\dot{\phi}_s$, gives:

$$\dot{v}_{2s-1} = (v_{2s-1}/r_s) (u_s r_s + r_s E_j \, a_{sj} \, r_{j2}) - v_{2s} (w_s + E_j \, b_{sj} \, r_{j2}),$$

and since $r_s^2 = v_{2s-1}^2 + v_{2s}^2$ ,

$$\dot{v}_{2s-1} = u_s \, v_{2s-1} - w_s \, v_{2s} + \overset{n/2}{E_j} [v_{2s-1} \, a_{sj} - v_{2s} \, b_{sj}] \, (v_{2j-1}^2 + v_{2j}^2)$$

Similarly,

$$\dot{v}_{2s} = u_s \, v_{2s} + w_s \, v_{2s-1} + \overset{n/2}{E_j} [v_{2s} \, a_{sj} + v_{2s-1} \, b_{sj}] \, (v_{2j-1}^2 + v_{2j}^2).$$

Setting the $b_{sj} = 0$ for simplicity, choosing $u_s = -\tau + 1$ to get a standard network form, and reindexing $i,j = 1,2,\ldots,n$ , we get the Cartesian equivalent of the polar normal form equations.

$$\dot{v}_i = -\tau \, v_i + \overset{n}{E_j} \, J_{ij} \, v_j + v_i \overset{n}{E_j} \, A_{ij} \, v_j^2$$

Here J is a matrix containing 2x2 blocks along the diagonal of the local couplings of the linear terms of each pair of the previous equations $v_{2s-1}$ , $v_{2s}$ , with $-\tau$ separated out of the diagonal terms. The matrix A has 2x2 blocks of identical coefficients $a_{sj}$ of the nonlinear terms from each pair.

$$J = \begin{bmatrix} 1 & -w_1 & & & \\ w_1 & 1 & & & \\ & & 1 & -w_2 & \\ & & w_2 & 1 & \\ & & & & \ddots \end{bmatrix} \qquad A = \begin{bmatrix} a_{11} & a_{11} & a_{12} & a_{12} \\ a_{11} & a_{11} & a_{12} & a_{12} \\ a_{21} & a_{21} & a_{22} & a_{22} \\ a_{21} & a_{21} & a_{22} & a_{22} \\ & & & & \ddots \end{bmatrix}$$

**Learning Transformation - Linear Term**
Second; J is the canonical form of a real matrix with complex
conjugate eigenvalues, where the conjugate pairs appear in
blocks along the diagonal as shown. The Cartesian normal form
equations describe the interaction of these linearly uncoupled
complex modes due to the coupling of the nonlinear terms. We
can interpret the normal form equations as network equations
in eigenvector (or "memory") coordinates, given by some diag-
onalizing transformation P, containing those eigenvectors as
its columns, so that $J = P^{-1} T P$. Then it is clear that T may
instead be determined by the reverse projection $T = P J P^{-1}$
back into network coordinates, if we start with desired eigen-
vectors and eigenvalues. We are free to choose as columns in
P, the real and imaginary vectors $[\underline{x}^s \cos \underline{\theta}^s , \underline{x}^s \sin \underline{\theta}^s]$ of the
cycles $\underline{x}^s e^{i\theta s}$ of any linearly independent set S of patterns
to be learned. If we write the matrix expression for the proj-
ection in component form, we recover the expression given in
the theorem for $T_{ij}$,

$$T_{ij} = E_{mn} P_{im} J_{mn} P^{-1}{}_{nj} .$$

**Nonlinear Term Projection**
The nonlinear terms are transformed as well, but the expres-
sion cannot be easily written in matrix form. Using the com-
ponent form of the transformation,

$$x_i = E_j P_{ij} v_j \quad , \quad \dot{x}_i = E_j P_{ij} \dot{v}_j \quad , \quad v_j = E_k P^{-1}{}_{jk} x_k \quad , \quad \text{and}$$

substituting into the Cartesian normal form, gives:

$$\dot{x}_i = (-\tau+1) E_j P_{ij} (E_k P^{-1}{}_{jk} x_k) + E_j P_{ij} E_k J_{jk} (E_l P^{-1}{}_{kl} x_l)$$

$$+ E_j P_{ij} (E_k P^{-1}{}_{jk} x_k) E_l A_{jl} (E_m P^{-1}{}_{lm} x_m) (E_n P^{-1}{}_{ln} x_n)$$

Rearranging the orders of summation gives,

$$\dot{x}_i = (-\tau+1) E_k (E_j Pi_j P^{-1}{}_{jk}) x_k + E_l (E_k E_j P_{ij} J_{jk} P^{-1}{}_{kl}) x_l$$

$$+ E_n E_m E_k (E_l E_j P_{ij} P^{-1}{}_{jk} A_{jl} P^{-1}{}_{lm} P^{-1}{}_{ln}) x_k x_m x_n$$

Finally, performing the bracketed summations and relabeling
indices gives us the network of the theorem,

$$\dot{x}_i = - \tau x_i + E_j T_{ij} x_j + E_{jkl} T_{ijkl} x_j x_k x_l$$

with the expression for the tensor of the nonlinear term,

$$T_{ijkl} = E_{mn} \, P_{im} \, A_{mn} \, P^{-1}_{mj} \, P^{-1}_{nk} \, P^{-1}_{nl} \qquad\qquad Q.E.D.$$

## LEARNING RULE EXTENSIONS

This is the core of the mathematical story, and it may be extended in many ways. When the columns of P are orthonormal, then $P^{-1} = P^T$, and the formula above for the linear network coupling becomes $T = PJP^T$. Then, for complex eigenvectors,

$$T_{ij} = E_s \, x_i^s \, x_j^s \, [\cos(\theta_i^s - \theta_j^s) + w_s \sin(\theta_i^s - \theta_j^s)].$$

This is now a local, additive, incremental learning rule for synapse ij, and the system can be truly self-organizing because the net can modify itself based on its own activity. Between units of equal phase, or when $\theta_i^s = \theta_j^s = 0$ for a static pattern, this reduces to the usual Hebb rule.

In a similar fashion, the learning rule for the higher order nonlinear terms becomes a multiple periodic outer product rule when the matrix A is chosen to have a simple form. Given our present ignorance of the full biophysics of intracellular processing, it is not entirely impossible that some dimensionality of the higher order weights in the mathematical network could be implemented locally within the cells of a biological network, using the information available on the primary lines given by the linear connections discussed above. When the A matrix is chosen to have uniform entries $A_{ij} = c$ for all its off-diagonal 2 x 2 blocks, and uniform entries $A_{ij} = c - d$ for the diagonal blocks, then,

$$\begin{aligned} T_{ijkl} = c\delta_{ij}\delta_{kl} - d E_s^{n/2} \, x_i^s \, x_j^s \, x_k^s \, x_l^s \, [&\cos\theta_i^s \cos\theta_j^s \cos\theta_k^s \cos\theta_l^s \\ + &\sin\theta_i^s \sin\theta_j^s \cos\theta_k^s \cos\theta_l^s + \cos\theta_i^s \cos\theta_j^s \sin\theta_k^s \sin\theta_l^s \\ + &\sin\theta_i^s \sin\theta_j^s \sin\theta_k^s \sin\theta_l^s]. \end{aligned}$$

This reduces to the multiple outer product

$$T_{ijkl} = c \, \delta_{ij} \, \delta_{kl} - d \, E_s^{n/2} \, x_i^s \, x_j^s \, x_k^s \, x_l^s \, , \text{ for static patterns.}$$

The network architecture generated by this learning rule is

$$\dot{x}_i = -\tau x_i + E_j \, T_{ij} \, x_j + c \, x_i \, E_j \, x_j^2 - d \, x_i \, E_{jkl} \, T_{ijkl} \, x_j \, x_k \, x_l \, .$$

This reduces to an architecture without higher order correlations in the case that we choose a completely uniform A matrix ($A_{ij} = c$, for all i,j). Then

$$\dot{x}_i = -\tau x_i + E_j \, T_{ij} \, x_j + c \, x_i \, E_j \, x_j^2$$

This network has fixed points on the axes of the normal form as always, but the stability condition is not satisfied since the diagonal normal form coefficients are equal, not less, than the remaining A matrix entries. In (Baird 89) we describe how clamped input (inspiration) can break this symmetry and make the nearest stored pattern be the only attractor.

All of the above results hold as well for networks with sigmoids, provided their coupling is such that they have a Taylor's expansion which is equal to the above networks up to third order. The results then hold only in the neighborhood of the origin for which the truncated expansion is accurate. The expected performance of such systems has been verified in simulations.

## Acknowledgements
Supported by AFOSR-87-0317. I am very grateful for the support of Walter Freeman and invaluable assistance of Morris Hirsch.

## References
B. Baird.   Bifurcation Theory Methods For Programming Static or Periodic Attractors and Their Bifurcations in Dynamic Neural Networks.   Proc. IEEE Int. Conf. Neural Networks, San Diego, Ca.,pI-9, July(1988).

B. Baird.   Bifurcation Theory Approach to Vectorfield Programming for Periodic Attractors.   Proc. INNS/IEEE Int. Conf. on Neural Networks. Washington D.C., June(1989).

W. J. Freeman & B. Baird.   Relation of Olfactory EEG to Behavior: Spatial Analysis.   Behavioral Neuroscience (1986).

W. J. Freeman & B. W. van Dijk.   Spatial Patterns of Visual Cortical EEG During Conditioned Reflex in a Rhesus Monkey. Brain Research, 422, p267(1987).

C. M. Grey and W. Singer. Stimulus Specific Neuronal Oscillations in Orientation Columns of Cat Visual Cortex. PNAS. In Press(1988).

Z. Li & J.J. Hopfield.   Modeling The Olfactory Bulb. Biological Cybernetics. Submitted(1989).
